# Blind Motion Deblurring Using Image Statistics

**Anat Levin**[*]
School of Computer Science and Engineering
The Hebrew University of Jerusalem

## Abstract

We address the problem of blind motion deblurring from a single image, caused by a few moving objects. In such situations only part of the image may be blurred, and the scene consists of layers blurred in different degrees. Most of of existing blind deconvolution research concentrates at recovering a single blurring kernel for the entire image. However, in the case of different motions, the blur cannot be modeled with a single kernel, and trying to deconvolve the entire image with the same kernel will cause serious artifacts. Thus, the task of deblurring needs to involve segmentation of the image into regions with different blurs.

Our approach relies on the observation that the statistics of derivative filters in images are significantly changed by blur. Assuming the blur results from a constant velocity motion, we can limit the search to one dimensional box filter blurs. This enables us to model the expected derivatives distributions as a function of the width of the blur kernel. Those distributions are surprisingly powerful in discriminating regions with different blurs. The approach produces convincing deconvolution results on real world images with rich texture.

## 1 Introduction

Motion blur is the result of the relative motion between the camera and the scene during image exposure time. This includes both camera and scene objects motion. As blurring can significantly degrade the visual quality of images, photographers and camera manufactures are frequently searching for methods to limit the phenomenon. One solution that reduces the degree of blur is to capture images using shorter exposure intervals. This, however, increases the amount of noise in the image, especially in dark scenes. An alternative approach is to try to remove the blur off-line. Blur is usually modeled as a linear convolution of an image with a blurring kernel, also known as the point spread function (or PSF). Image deconvolution is the process of recovering the unknown image from its blurred version, given a blurring kernel. In most situations, however, the blurring kernel is unknown as well, and the task also requires the estimation of the underlying blurring kernel. Such a process is usually referred to as *blind deconvolution*.

Most of the existing blind deconvolution research concentrates at recovering a single blurring kernel for the entire image. While the uniform blur assumption is valid for a restricted set of camera motions, it's usually far from being satisfying when the scene contains several objects moving independently. Existing deblurring methods which handle different motions usually rely on multiple frames. In this work, however, we would like to address blind multiple motions deblurring using a single frame.

The suggested approach is fully automatic, under the following two assumptions. The first assumption is that the image consists of a small number of blurring layers with the same blurring kernel within each layer. Most of the examples in this paper include a single blurred object and an unblurred background. Our second simplifying assumption is that the motion is in a single direction

---

[*]Current address: MIT CSAIL, alevin@csail.mit.edu

and that the motion velocity is constant, such as in the case of a moving vehicle captured by a static camera. As a result, within each blurred layer, the blurring kernel is a simple one dimensional box filter, so that the only unknown parameters are the blur direction and the width of the blur kernel.

Deblurring different motions requires the *segmentation* of the image into layers with different blurs as well as the reconstruction of the blurring kernel in each layer. While image segmentation is an active and challenging research area which utilizes various low level and high level cues, the only segmentation cue used in this work is the degree of blur. In order to discriminate different degrees of blur we use the statistics of natural images. Our observation is that statistics of derivatives responses in images are significantly changed as a result of blur, and that the expected statistics under different blurring kernels can be modeled. Given a model of the derivatives statistics under different blurring kernels our algorithm searches for a mixture model that will best describe the distribution observed in the input image. This results in a set of 2 (or some other small number) blurring kernels that were used in the image. In order to segment the image into blurring layers we measure the likelihood of the derivatives in small image windows, under each model. We then look for a smooth layers assignment that will maximize the likelihood in each local window.

## 1.1    Related work

Blind deconvolution is an extensive research area. Research about blind deconvolution given a single image, usually concentrate at cases in which the image is uniformly blurred. A summary and analysis of many deconvolution algorithms can be found in [14]. Early deblurring methods treated blurs that can be characterized by a regular pattern of zeros in the frequency domain such as box filter blurs [26]. This method is known to be very sensitive to noise. Even in the noise free case, box filter blurs can not be identified in the frequency domain if different blurs are present. More recent methods are making other assumptions about the image model. This includes an autoregressive process [22], spatial isotropy [28], power low distributions [8, 20], and piecewise-smoothness edges modeling [3]. In a creative recent research which inspired our approach, Fergus et al [12] use the statistics of natural images to estimate the blurring kernel (again, assuming a uniform blur). Their approach searches for the max-marginal blurring kernel and a deblurred image, using a prior on derivatives distribution in an unblurred image. They address more than box filters, and present impressing reconstructions of complex blurring kernels. Our approach also relies on natural images statistics, but it takes the opposite direction: search for a kernel that will bring the unblurred distribution close to the observed distribution. Thus, in addition to handling non uniform blurs, our approach avoids the need to estimate the unblurred image in every step.

In [10], Elder and Zucker propose a scale space approach for estimating the scale of an edge. As the edge's scale provides some measure of blur this is used for segmenting an image into a focus and out of focus layers. The approach was demonstrated on a rather piecewise constant image, unlike the rich texture patterns considered in this paper. In [4], blind restoration of spatially-varying blur was studied in the case of astronomical images, which have statistics quite different from the natural scenes addressed in this paper.

Other approaches to motion deblurring include hardware approaches [6, 17, 7], and using multiple frames to estimate blur, e.g. [5, 21, 29].

Another related subject is the research on depth from focus or depth from defocus (see [9, 11] to name a few), in which a scene is captured using multiple focus settings. As a scene point focus is a function of its depth, the relative blur is used to estimate depth information. Again, most of this research relies on more than a single frame.

Recent work in computer vision applied natural images priors for a variety of applications like denoising [25, 24], super resolution [27], video matting [2], inpainting [16] and reflections decomposition [15].

## 2    Image statistics and blurring

Figure 1(a) presents an image of an outdoor scene, with a passing bus. The bus is blurred horizontally as a result of the bus motion. In fig 1(b) we plot the log histogram of the vertical derivatives of this image, and the horizontal derivatives within the blurred area (marked with a rectangle). As can be

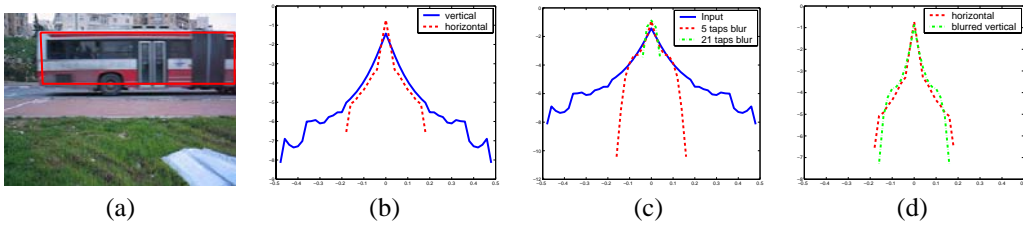

|      (a)      |      (b)      |      (c)      |      (d)      |

Figure 1: Blurred versus unblurred derivatives histograms. (a) Input image. (b) Horizontal derivatives within the blurred region versus vertical derivatives in the entire image. (c) Simulating different blurs in the vertical direction. (d) Horizontal derivatives within the blurred region matched with blurred verticals (4 tap blur).

seen, the blur changes the shape of the histogram significantly. This suggests that the statistics of derivative filters responses can be used for detecting blurred image areas.

How does the degree of blur affects the derivatives histogram? To answer this question we simulate histograms of different blurs. Let $f_k$ denote the horizontal box kernel of size $1 \times k$ (that is, all entries of $f_k$ equal $1/k$). We convolve the image with the kernels $f_k^T$ (where k runs from 1 to 30) and compute the vertical derivatives distributions:

$$p_k \propto hist(d_y * f_k^T * I) \tag{1}$$

where $d_y = [1 - 1]^T$. Some of those log histograms are plotted in fig 1(c). As the size of the blurring kernel changes the derivatives distribution, we would also like to use the histograms for determining the degree of blur. For example, as illustrated in fig 1(d), we can match the distribution of vertical derivatives in the blurred area, and $p_4$, the distribution of horizontal derivatives after blurring with a 4 tap kernel.

## 2.1  Identifying blur using image statistics

Given an image, the direction of motion blur can be selected as the direction with minimal derivatives variation, as in [28]. For the simplicity of the derivation we will assume here that the motion direction is horizontal, and that the image contains a single blurred object plus an unblurred background. Our goal is to determine the size of the blur kernel. That is, to recover the filter $f_k$ which is responsible for the blur observed in the image. For that we compute the histogram of horizontal derivatives in the image. However, not all the image is blurred. Therefore, without segmenting the blurred areas there is no single blurring model $p_k$ that will describe the observed histogram. Instead, we try to describe the observed histogram with a mixture model. We define the log-likelihood of the derivatives in a window with respect to each of the blurring models as:

$$\ell_k(i) = \sum_{j \in W_i} \log p_k(I_x(j)) \tag{2}$$

Where $I_x(j)$ is the horizontal derivative in pixel $j$, and $W_i$ is a window around pixel $i$. Thus, $\ell_k(i)$ measures how well the $i$'th window is explained by a $k$-tap blur.

For an input image $I$ and a given pair of kernels, we can measure the data log-likelihood by associating each window with the maximum likelihood kernel:

$$L(I|f_{k_1}, f_{k_2}) = \sum_{i \in I} max(\ell_{k_1}(i), \ell_{k_2}(i)) \tag{3}$$

We search for a blurring model $p_{k_0}$ such that, when combined with the model $p_1$ (derivatives of the unblurred image), will maximize the log-likelihood of the observed derivatives:

$$k_0 = \arg \max_k L(I|f_1, f_k) \tag{4}$$

One problem we need to address in defining the likelihoods is the fact that uniform areas, or areas with pure horizontal edges (the aperture problem) don't contain any information about the blur.

On the other hand, uniform areas receive the highest likelihoods from wide blur kernels (since the derivatives distribution for wide kernels is more concentrated around zero, as can be observed in figure 1(c)). When the image consists of large uniform areas, this bias the likelihood toward wider blur kernels. To overcome this, we start by scanning the image with a simple edge detector and keep only windows with significant vertical edges. In order to make our model consistent, when building the blurred distribution models $p_k$ (eq 1), we also take into account only pixels within a window around a vertical edge.

Note that since we deal here with one dimensional kernels, we can estimate the expected blurred histogram $p_k$ (eq 1) from the perpendicular direction of the same image.

## 2.2  Segmenting blur layers

Once the blurring kernel $f_k$ has been found, we can use it to deconvolve the image, as in fig 2(b). While this significantly improves the image in the blurred areas, serious artifacts are observed in the background. Therefore, in addition to recovering the blurring kernel, we need to segment the image into blurred and unblurred layers. We look for a smooth segmentation that will maximize the likelihood of the derivatives in each region. We define the energy of a segmentation as:

$$E(x) = \sum_i -\ell(x(i), i) + \sum_{<ij>} e_{ij} |x(i) - x(j)| \tag{5}$$

where $\ell(x(i), i) = \ell_1(i)$ for $x(i) = 0$ and $\ell(x(i), i) = \ell_k(i)$ for $x(i) = 1$, $< i, j >$ are neighboring image pixels, and $e_{ij}$ is a smoothness term:

$$e_{ij} = \lambda + \nu(|I(i) - I^{-f_k}(i)| + |I(j) - I^{-f_k}(j)|) \tag{6}$$

Here $I^{-f_k}$ denotes the deconvolved image. The smoothness term is combined from two parts. The first is just a constant penalty for assigning different labels to neighboring pixels, thus preferring smooth segmentations. The second part encodes the fact that it is cheaper to cut the image in places where there is no visual seam between the original and the deconvolved images (e.g. [1]).

Given the local likelihood scores and the energy definition, we would like to find the minimal energy segmentation. This reduces to finding a min-cut in a graph. Given the segmentation mask $x$ we convolve it with a Gaussian filter to obtain a smoother seam. The final restored image is computed as:

$$R(i) = x(i)I^{-f_k}(i) + (1 - x(i))I(i) \tag{7}$$

## 3  Results

To compute a deconvolved image $I^{-f_k}$ given the blurring kernel, we follow [12] in using the matlab implementation (`deconvlucy`) of the Richardson-Lucy deconvolution algorithm [23, 18].

Figure 2 presents results for several example images. For the doll example the image was segmented into 3 blurring layers. The examples of figure 2 and additional results are available in a high resolution in the supplementary material. The supplementary file also includes examples with non horizontal blurs. To determine the blur direction in those images we select the direction with minimal derivatives variation, as in [28]. This approach wasn't always robust enough.

For each image we show what happens if the segmentation is ignored and the entire image is deconvolved with the selected kernel (for the doll case the wider kernel is shown). While this improves the result in the blurred area, strong artifacts are observed in the rest of the image. In comparison, the third row presents the restored images computed from eq 7 using the blurring layers segmentation. We also show the local MAP labeling of the edges. White pixels are ones for which an unblurred model receives a higher likelihood, that is $\ell_1(i) > \ell_k(i)$, and for gray pixels $\ell_1(i) < \ell_k(i)$ (for the doll case there are 3 groups, defined in a similar way). The last row presents the segmentation contour. The output contour does not perfectly align with image edges. This is because our goal in the segmentation selection is to produce visually plausible results. The smoothness term of our energy (eq 6) does not aim to output an accurate segmentation, and it does not prefer to align segmentation edges with image edges. Instead it searches for a cut that will make the seam between the layers unobservable.

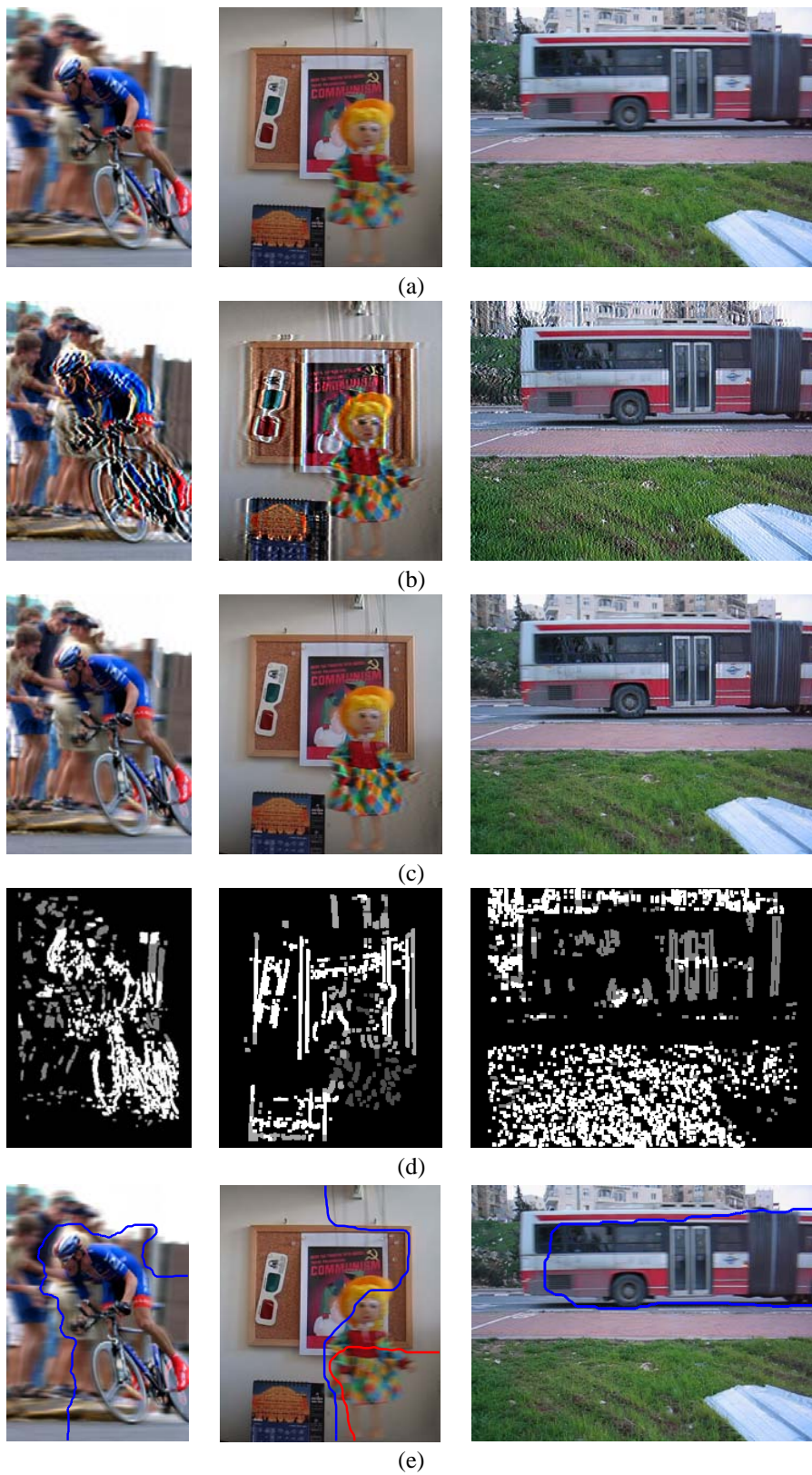

Figure 2: Deblurring Results. (a)Input image. (b)Applying the recovered kernel on the entire image. (c)Our result. (d)Local classification of windows. (e)Segmentation contour

The recovered blur sizes for those examples were 12 pixels for the bicycles image and 4 pixels for the bus. For the doll image a 9 pixels blur was identified in the skirt segment and a 2 pixels blur in the doll head. We note that while recovering big degrees of blur as in the bicycles example is visually more impressing, discriminating small degrees of blur as in the bus example is more challenging from the statistical aspect. This is because the derivatives distributions in the case of small blurs are much more similar to the distributions of unblurred images.

For the bus image the size of the blur kernel found by our algorithm was 4 pixels. To demonstrate the fact that this is actually the true kernel size, we show in figure 3 the deconvolution results with a 3-tap filter and with a 5-tap filter. Stronger artifacts are observed in each of those cases.

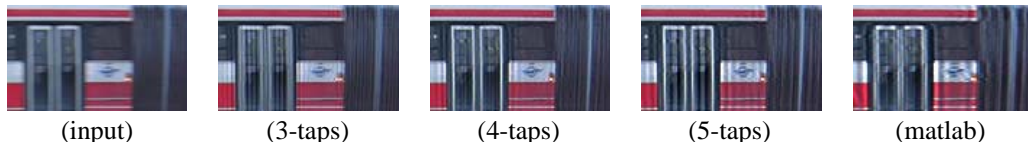

| (input) | (3-taps) | (4-taps) | (5-taps) | (matlab) |

Figure 3: Deconvolving the bus image using different filters. The 4-tap filter selected by our algorithm yields best results

Next, we consider several simple alternatives to some of the algorithm parts. We start by investigating the need in segmentation and then discuss the usage of the image statistics.

**Segmentation:** As demonstrated in fig 2(b) deconvolving the entire image with the same kernel damages the unblurred parts. One obvious solution is to divide the image into regions and match a separate blur kernel to each region. As demonstrated by fig 2(d), even if we limit the kernel choice in each local window to a small set of 2-3 kernels, the local decision could be wrong. For all the examples in this paper we used $15 \times 35$ windows. There is some tradeoff in selecting a good window size. While likelihood measure based on a big window is more reliable, such a window might cover regions from different blurring layers. Another alternative is to brake the image into segments using an unsupervised segmentation algorithm, and match a kernel to each segment. The fact that blur changes the derivatives distributions also suggests that it might be captured as a kind of texture cue. Therefore, it's particularly interesting to try segmenting the image using texture affinities (e.g. [13, 19]). However, as this is an unsupervised segmentation process which does not take into account the grouping goal, it's hard to expect it to yield exactly the blurred layers. Fig 4(b) presents segmentation results using the Ncuts framework of [19]. The output over-segments blur layers, while merging parts of blurred and unblurred objects. Unsurprisingly, the recovered kernels are wrong.

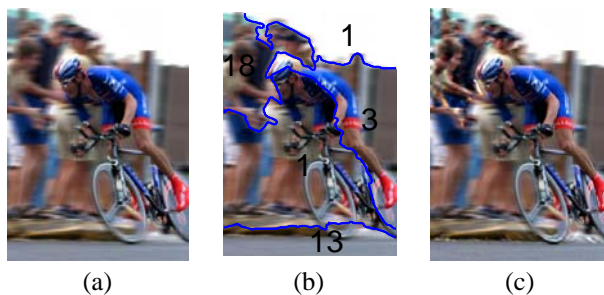

| (a) | (b) | (c) |

Figure 4: Deblurring using unsupervised segmentation. (a) Input. (b) Unsupervised segmentation and the width of the kernel matched to each segment. (c) Result from deblurring each segment independently.

**Image statistics:** We move to evaluating the contribution of the image statistics. To do that independently of the segmentation, we manually segmented the bus and applied the matlab blind deconvolution function (`deconvblind`), initialized with a $1 \times 7$ box kernel. Strong artifacts were introduced as shown in the last column of fig 3.

The algorithm results also depend on the actual histograms used. Derivatives histograms of different natural images usually have common characteristics such as the heavy tail structure. Yet, the histogram structure of different images is not identical, and we found that trying to deblur one image using the statistics of a different image doesn't work that well. For example, figure 5 shows the result of deblurring the bus image using the bicycles image statistics. The selected blur in this case was a 6-tap kernel, but deblurring the image with this kernel introduces artifacts. The classification of pixels into layers using this model is wrong as well. Our solution was to work on each image using the vertical derivatives histograms from the same image. This isn't an optimal solution as when the image is blurred horizontally some of the vertical derivatives are degraded as well. Yet, it provided better results than using histograms obtained from different images.

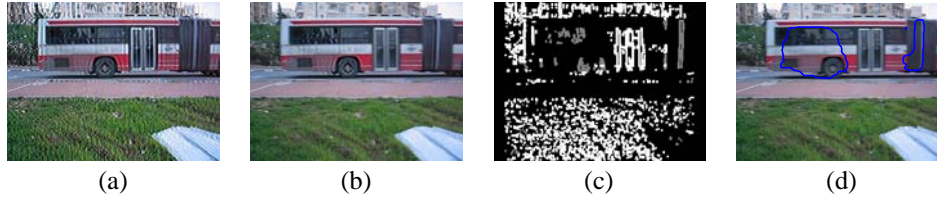

| (a) | (b) | (c) | (d) |

Figure 5: Deblurring the bus image using the bicycles image statistics. (a) Applying the recovered kernel on the entire image. (b) Deblurring result. (c) Local classification of windows. (d) Segmentation contour.

**Limitations:** Our algorithm uses simple derivatives statistics and the power of such statistics is somewhat surprising. Yet, the algorithm might fail. One failure source is blurs which can't be described as a box filter, or failures in identifying the blur direction. Even when this isn't the case, the algorithm may fail to identify the correct blur size or it may not infer the correct segmentation. Figure 6 demonstrate a failure. In this case the algorithm preferred a model explaining the bushes texture instead of a model explaining the car blur. The bushes area consists of many small derivatives which are explained better by a small blur model than by a no-blur model. On the other hand, the car consists of very few vertical edges. As a result the algorithm selected a 6-pixels blur model. This model might increase the likelihood of the bushes texture and the noise on the road, but it doesn't remove the blur of the car.

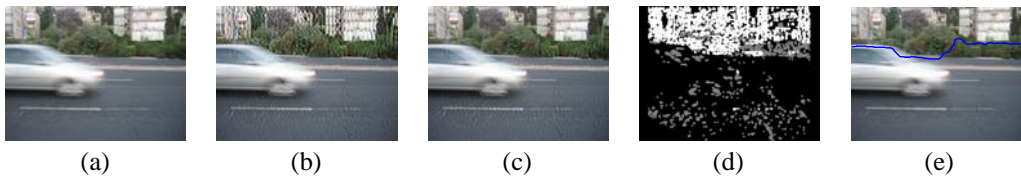

| (a) | (b) | (c) | (d) | (e) |

Figure 6: Deblurring failure. (a) Input. (b) Applying the recovered kernel (6-taps) on the entire image. (c) Deblurring result. (d) Local classification. (e) Segmentation contour.

## 4 Discussion

This paper addresses the problem of blind motion deconvolution without assuming that the entire image undergone the same blur. Thus, in addition to recovering an unknown blur kernel, we segment the image into layers with different blurs. We treat this highly challenging task using a surprisingly simple approach, relying on the derivatives distribution in blurred images. We model the expected derivatives distributions under different degrees of blur, and those distributions are used for detecting different blurs in image windows.

The box filters model used in this work is definitely limiting, and as pointed out by [12, 6], many blurring patterns observed in real images are more complex.

A possible future research direction is to try to develop stronger statistical models which can include stronger features in addition to the simple first order derivatives. Stronger models might enable us to identify a wider class of blurring kernels rather than just box filters. Particularly, they could provide

a better strategy for identifying the blur direction. A better model might also avoid the need to detect vertical edges and artificially limit the model to windows around edges.

In future work, it will also be interesting to try to detect different blurs without assuming a small number of blurring layers. This will require estimating the blurs in the image in a continues way, and might also provide a depth from focus algorithm that will work on a single image.

## References

[1] A. Agarwala et al. Interactive digital photomontage. *SIGGRAPH*, 2004.

[2] N. Apostoloff and A. Fitzgibbon. Bayesian image matting using learnt image priors. In *CVPR*, 2005.

[3] L. Bar, N. Sochen, and N. Kiryati. Variational pairing of image segmentation and blind restoration. In *ECCV*, 2004.

[4] J. Bardsley, S. Jefferies, J. Nagy, and R. Plemmons. Blind iterative restoration of images with spatially-varying blur. *Optics Express*, 2006.

[5] B. Bascle, A. Blake, and A. Zisserman. Motion deblurring and superresolution from an image sequence. In *ECCV*, 1996.

[6] M. Ben-Ezra and S. K. Nayar. Motion-based motion deblurring. *PAMI*, 2004.

[7] 2006 Canon Inc. What is optical image stabilizer? `http://www.canon.com/bctv/faq/optis.html`.

[8] J. Caron, N. Namazi, and C. Rollins. Noniterative blind data restoration by use of an extracted filter function. *Applied Optics*, 2002.

[9] T. Darrell and K. Wohn. Pyramid based depth from focus. In *CVPR*, 1988.

[10] J. H. Elder and S. W. Zucker. Local scale control for edge detection and blur estimation. *PAMI*, 1998.

[11] P. Favaro, S. Osher, S. Soatto, and L.A. Vese. 3d shape from anisotropic diffusion. In *CVPR*, 2003.

[12] R. Fergus et al. Removing camera shake from a single photograph. *SIGGRAPH*, 2006.

[13] T. Hofmann, J. Puzicha, and J. M. Buhmann. Unsupervised texture segmentation in a deterministic annealing framework. *PAMI*, 1998.

[14] D. Kundur and D. Hatzinakos. Blind image deconvolution. *IEEE Signal Processing Magazine*, 1996.

[15] A. Levin and Y. Weiss. User assisted separation of reflections from a single image using a sparsity prior. In *ECCV*, 2004.

[16] A. Levin, A. Zomet, and Y. Weiss. Learning how to inpaint from global image statistics. In *ICCV*, 2003.

[17] X. Liu and A. Gamal. Simultaneous image formation and motion blur restoration via multiple capture. In *Int. Conf. Acoustics, Speech, Signal Processing*, 2001.

[18] L. Lucy. Bayesian-based iterative method of image restoration. *Journal of Ast.*, 1974.

[19] J. Malik, S. Belongie, T. Leung, and J. Shi. Contour and texture analysis for image segmentation. In *Perceptual Organization for artificial vision systems*. Kluwer Academic, 2000.

[20] R. Neelamani, H. Choi, and R. Baraniuk. Forward: Fourier-wavelet regularized deconvolution for ill-conditioned systems. *IEEE Trans. on Signal Processing*, 2004.

[21] A. Rav-Acha and S. Peleg. Two motion-blurred images are better than one. *Pattern Recognition Letters*, 2005.

[22] S. Reeves and R. Mersereau. Blur identification by the method of generalized cross-validation. *Transactions on Image Processing*, 1992.

[23] W. Richardson. Bayesian-based iterative method of image restoration. *J. of the Optical Society of America*, 1972.

[24] S. Roth and M.J. Black. Fields of experts: A framework for learning image priors. In *CVPR*, 2005.

[25] E. P. Simoncelli. Statistical modeling of photographic images. In *Handbook of Image and Video Processing*, 2005.

[26] T. Stockham, T. Cannon, and R. Ingebretsen. Blind deconvolution through digital signal processing. *IEEE*, 1975.

[27] M. F. Tappen, B. C. Russell, and W. T. Freeman. Exploiting the sparse derivative prior for super-resolution and image demosaicing. *SCTV*, 2003.

[28] Y. Yitzhaky, I. Mor, A. Lantzman, and N.S. Kopeika. Direct method for restoration of motion blurred images. *JOSA-A*, 1998.

[29] M. Shiand J. Zheng. A slit scanning depth of route panorama from stationary blur. In *Proc. IEEE Conf. Comput. Vision Pattern Recog.*, 2005.
